# Bayesian Hierarchical Reinforcement Learning

**Feng Cao**
Department of EECS
Case Western Reserve University
Cleveland, OH 44106
fxc100@case.edu

**Soumya Ray**
Department of EECS
Case Western Reserve University
Cleveland, OH 44106
sray@case.edu

## Abstract

We describe an approach to incorporating Bayesian priors in the MAXQ framework for hierarchical reinforcement learning (HRL). We define priors on the primitive environment model and on task pseudo-rewards. Since models for composite tasks can be complex, we use a mixed model-based/model-free learning approach to find an optimal hierarchical policy. We show empirically that (i) our approach results in improved convergence over non-Bayesian baselines, (ii) using both task hierarchies and Bayesian priors is better than either alone, (iii) taking advantage of the task hierarchy reduces the computational cost of Bayesian reinforcement learning and (iv) in this framework, task pseudo-rewards can be learned instead of being manually specified, leading to hierarchically optimal rather than recursively optimal policies.

## 1 Introduction

Reinforcement learning (RL) is a well known framework that formalizes decision making in unknown, uncertain environments. RL agents learn policies that map environment states to available actions while optimizing some measure of long-term utility. While various algorithms have been developed for RL [1], and applied successfully to a variety of tasks [2], the standard RL setting suffers from at least two drawbacks. First, it is difficult to scale standard RL approaches to large state spaces with many factors (the well-known "curse of dimensionality"). Second, vanilla RL approaches do not incorporate prior knowledge about the environment and good policies.

*Hierarchical reinforcement learning* (HRL) [3] attempts to address the scaling problem by simplifying the overall decision making problem in different ways. For example, one approach introduces macro-operators for sequences of primitive actions. Planning at the level of these operators may result in simpler policies [4]. Another idea is to decompose the task's overall value function, for example by defining task hierarchies [5] or partial programs with choice points [6]. The structure of the decomposition provides several benefits: first, for the "higher level" subtasks, policies are defined by calling "lower level" subtasks (which may themselves be quite complex); as a result policies for higher level subtasks may be expressed compactly. Second, a task hierarchy or partial program can impose constraints on the space of policies by encoding knowledge about the structure of good policies and thereby reduce the search space. Third, learning within subtasks allows *state abstraction*, that is, some state variables can be ignored because they do not affect the policy within that subtask. This also simplifies the learning problem.

While HRL attempts to address the scalability issue, it does not take into account probabilistic prior knowledge the agent may have about the task. For example, the agent may have some idea about where high/low utility states may be located and what their utilities may be, or some idea about the approximate shape of the value function or policy. *Bayesian reinforcement learning* addresses this issue by incorporating priors on models [7], value functions [8, 9] or policies [10]. Specifying good

priors leads to many benefits, including initial good policies, directed exploration towards regions of uncertainty, and faster convergence to the optimal policy.

In this paper, we propose an approach that incorporates Bayesian priors in hierarchical reinforcement learning. We use the MAXQ framework [5], that decomposes the overall task into subtasks so that value functions of the individual subtasks can be combined to recover the value function of the overall task. We extend this framework by incorporating priors on the primitive environment model and on task pseudo-rewards. In order to avoid building models for composite tasks (which can be very complex), we adopt a mixed model-based/model-free learning approach. We empirically evaluate our algorithm to understand the effect of the priors in addition to the task hierarchy. Our experiments indicate that: (i) taking advantage of probabilistic prior knowledge can lead to faster convergence, even for HRL, (ii) task hierarchies and Bayesian priors can be complementary sources of information, and using both sources is better than either alone, (iii) taking advantage of the task hierarchy can reduce the computational cost of Bayesian RL, which generally tends to be very high, and (iv) task pseudo-rewards can be learned instead of being manually specified, leading to automatic learning of hierarchically optimal rather than recursively optimal policies. In this way Bayesian RL and HRL are synergistic: Bayesian RL improves convergence of HRL and can learn hierarchy parameters, while HRL can reduce the significant computational cost of Bayesian RL.

Our work assumes the probabilistic priors to be given in advance and focuses on learning with them. Other work has addressed the issue of obtaining these priors. For example, one source of prior information is multi-task reinforcement learning [11, 12], where an agent uses the solutions of previous RL tasks to build priors over models or policies for future tasks. We also assume the task hierarchy is given. Other work has explored learning MAXQ hierarchies in different settings [13].

## 2 Background and Related Work

In the MAXQ framework, each composite subtask $T_i$ defines a semi-Markov decision process with parameters $\langle S_i, X_i, C_i, G_i \rangle$. $S_i$ defines the set of "non-terminal" states for $T_i$, where $T_i$ may be called by its parent. $G_i$ defines a set of "goal" states for $T_i$. The actions available within $T_i$ are described by the set of "child tasks" $C_i$. Finally, $X_i$ denotes the set of "relevant state variables" for $T_i$. Often, we unify the non-$S_i$ states and $G_i$ into a single "termination" predicate, $P_i$. An $(s, a, s')$ triple where $P_i(s)$ is false, $P_i(s')$ is true, $a \in C_i$, and the transition probability $P(s'|s, a) > 0$ is called an *exit* of the subtask $T_i$. A pseudo-reward function $\tilde{R}(s, a)$ can be defined over exits to express preferences over the possible exits of a subtask.

A hierarchical policy $\pi$ for the overall task is an assignment of a local policy to each SMDP $T_i$. A *hierarchically optimal policy* is a hierarchical policy that has the maximum expected reward. A hierarchical policy is said to be *recursively optimal* if the local policy for each subtask is optimal given that all its subtask policies are optimal. Given a task graph, model-free [5] or model-based [14] methods can be used to learn value functions for each task-subtask pair. In the model-free method, a policy is produced by maintaining a value and a *completion* function for each subtask. For a task $i$, the value $V(a, s)$ denotes the expected value of calling child task $a$ in state $s$. This is (recursively) estimated as the expected reward obtained while executing $a$. The completion function $C(i, s, a)$ denotes the expected reward obtained while *completing* $i$ after having called $a$ in $s$. The central idea behind MAXQ is that the value of $i$, $V(i, s)$, can be (recursively) decomposed in terms of $V(a, s)$ and $C(i, s, a)$. The model-based RMAXQ [14] algorithm extends RMAX [15] to MAXQ by learning models for all primitive and composite tasks. Value iteration is used with these models to learn a policy for each subtask. An optimistic exploration strategy is used together with a parameter $m$ that determines how often a transition or reward needs to be seen to be usable in the planning step.

In the MAXQ framework, pseudo-rewards must be manually specified to learn hierarchically optimal policies. Recent work has attempted to directly learn hierarchically optimal policies for ALisp partial programs, that generalize MAXQ task hierarchies [6, 16], using a model-free approach. Here, along with task value and completion functions, an "external" $Q$ function $Q_E$ is maintained for each subtask. This function stores the reward obtained after the parent of a subtask exits. A problem here is that this hurts state abstraction, since $Q_E$ is no longer "local" to a subtask. In later work [16], this is addressed by recursively representing $Q_E$ in terms of task value and completion functions, linked by conditional probabilities of parent exits given child exits. The conditional probabilities and recursive decomposition are used to compute $Q_E$ as needed to select actions.

Bayesian reinforcement learning methods incorporate probabilistic prior knowledge on models [7], value functions [8, 9], policies [10] or combinations [17]. One Bayesian model-based RL algorithm proceeds as follows. At each step, a distribution over model parameters is maintained. At each step, a model is sampled from this distribution (Thompson sampling [18, 19]). This model is then solved and actions are taken according to the policy obtained. This yields observations that are used to update the parameters of the current distribution to create a posterior distribution over models. This procedure is then iterated to convergence. Variations of this idea have been investigated; for example, some work converts the distribution over models to an empirical distribution over $Q$-functions, and produces policies by sampling from this distribution instead [7].

Relatively little work exists that attempts to incorporate probabilistic priors into HRL. We have found one preliminary attempt [20] that builds on the RMAX+MAXQ [14] method. This approach adds priors to each subtask model and performs (separate) Bayesian model-based learning for each subtask. [1] In our approach, we do not construct models for subtasks, which can be very complex in general. Instead, we only maintain distributions over primitive actions, and use a mixed model-based/model-free learning algorithm that is naturally integrated with the standard MAXQ learning algorithm. Further, we show how to learn pseudo-rewards for MAXQ in the Bayesian framework.

## 3  Bayesian MAXQ Algorithm

In this section, we describe our approach to incorporating probabilistic priors into MAXQ. We use priors over primitive models and pseudo-rewards. As we explain below, pseudo-rewards are value functions; thus our approach uses priors both on models and value functions. While such an integration may not be needed for standard Bayesian RL, it appears naturally in our setting.

We first describe our approach to incorporating priors on environment models alone (assuming pseudo-rewards are fixed). We do this following the Bayesian model-based RL framework. At each step we have a distribution over environment models (initially the prior). The algorithm has two main subroutines: the main BAYESIAN_MAXQ routine (Algorithm 1) and an auxiliary RECOMPUTE_VALUE routine (Algorithm 2). In this description, the value $V$ and completion $C$ functions are assumed to be global. At the start of each episode, the BAYESIAN_MAXQ routine is called with the $Root$ task and the initial state for the current episode. The MAXQ execution protocol is then followed, where each task chooses an action based on its current value function (initially random). When a primitive action is reached and executed, it updates the posterior over model parameters (Line 3) and its own value estimate (which is just the reward function for primitive actions). When a task exits and returns to its parent, the parent subsequently updates its completion function based on the current estimates of the value of the exit state (Lines 14 and 15). Note that in MAXQ, the value function of a composite task can be (recursively) computed using the completion functions of subtasks and the rewards obtained by executing primitive actions, so we do not need to separately store or update the value functions (except for the primitive actions where the value function is the reward). Finally, each primitive action maintains a count of how many times it has been executed and each composite task maintains a count of how many child actions have been taken.

When $k$ (an algorithm parameter) steps have been executed in a composite task, BAYESIAN_MAXQ calls RECOMPUTE_VALUE to re-estimate the value and completion functions (the check on $k$ is shown in RECOMPUTE_VALUE, Line 2). When activated, this function recursively re-estimates the value/completion functions for all subtasks of the current task. At the level of a primitive action, this simply involves resampling the reward and transition parameters from the current posterior over models. For a composite task, we use the MAXQ-Q algorithm (Table 4 in [5]). We run this algorithm for $Sim$ episodes, starting with the current subtask as the root, with the current pseudo-reward estimates (we explain below how these are obtained). This algorithm recursively updates the completion function of the task graph below the current task. Note that in this step, the subtasks with primitive actions use model-based updates. That is, when a primitive action is "executed" in such tasks, the currently sampled transition function (part of $\Theta$ in Line 5) is used to find the next state, and then the associated reward is used to update the completion function. This is similar to Lines 12, 14 and 15 in BAYESIAN_MAXQ, except that it uses the sampled model $\Theta$ instead of the

**Algorithm 1** BAYESIAN_MAXQ

---

**Input:** Task $i$, State $s$, Update Interval $k$, Simulation Episodes $Sim$
**Output:** Next state $s'$, steps taken $N$, cumulative reward $CR$

1: **if** $i$ is primitive **then**
2:     Execute $i$, observe $r$, $s'$
3:     Update current posterior parameters $\Psi$ using $(s, i, r, s')$
4:     Update current value estimate: $V(i, s) \leftarrow (1 - \alpha) \cdot V(i, s) + \alpha \cdot r$
5:     $Count(i) \leftarrow Count(i) + 1$
6:     **return** $(s', 1, r)$
7: **else**
8:     $N \leftarrow 0, CR \leftarrow 0, taskStack \leftarrow Stack()$ {$i$ is composite}
9:     **while** $i$ is not terminated **do**
10:        RECOMPUTE_VALUE$(i, k, Sim)$
11:        $a \leftarrow \epsilon$-greedy action from $V(i, s)$
12:        $\langle s', N_a, cr \rangle \leftarrow$ BAYESIAN_MAXQ$(a, s)$
13:        $taskStack.push(\langle a, s', N_a, cr \rangle)$
14:        $a^*_{s'} \leftarrow \arg\max_{a'} \left[ \tilde{C}(i, s', a') + V(a', s') \right]$
15:        $C(i, s, a) \leftarrow (1 - \alpha) \cdot C(i, s, a) + \alpha \cdot \gamma^{N_a} \left[ C(i, s', a^*_{s'}) + V(a^*_{s'}, s') \right]$
16:        $\tilde{C}(i, s, a) \leftarrow (1 - \alpha) \cdot \tilde{C}(i, s, a) + \alpha \cdot \gamma^{N_a} \left[ \tilde{R}(i, s') + \tilde{C}(i, s', a^*_{s'}) + V(a^*_{s'}, s') \right]$
17:        $s \leftarrow s', CR \leftarrow CR + \gamma^N \cdot cr, N \leftarrow N + N_a, Count(i) \leftarrow Count(i) + 1$
18:     **end while**
19:     UPDATE_PSEUDO_REWARD$(taskStack, \tilde{R}(i, s'))$
20:     **return** $(s', N, CR)$
21: **end if**

---

**Algorithm 2** RECOMPUTE_VALUE

---

**Input:** Task $i$, Update Interval $k$, Simulation Episodes $Sim$
**Output:** Recomputed value and completion functions for the task graph below and including $i$

1: **if** $Count(i) < k$ **then**
2:     **return**
3: **end if**
4: **if** $i$ is primitive **then**
5:     Sample new transition and reward parameters $\Theta$ from current posterior $\Psi$
6: **else**
7:     **for all** child tasks $a$ of $i$ **do**
8:        RECOMPUTE_VALUE$(a, k, Sim)$
9:     **end for**
10:     **for** $Sim$ episodes **do**
11:        $s \leftarrow$ random nonterminal state of $i$
12:        Run MAXQ-Q$(i, s, \Theta, \tilde{R})$
13:     **end for**
14: **end if**
15: $Count(i) \leftarrow 0$

---

real environment. After RECOMPUTE_VALUE terminates, a new set of value/completion functions are available for BAYESIAN_MAXQ to use to select actions.

Next we discuss task pseudo-rewards (PRs). A PR is a value associated with a subtask exit that defines how "good" that exit is for that subtask. The *ideal* PR for an exit is the expected reward under the hierarchically optimal policy after exiting the subtask, until the global task (Root) ends; thus the PR is a value function. This PR would enable the subtask to choose the "right" exit *in the context of* what the rest of the task hierarchy is doing. In standard MAXQ, these have to be set manually. This is problematic because it presupposes (quite detailed) knowledge of the hierarchically optimal policy. Further, setting the *wrong* PRs can result in non-convergence or highly suboptimal policies. Sometimes this problem is sidestepped simply by setting all PRs to zero, resulting in recursively optimal policies. However, it is easy to construct examples where a recursively optimal policy

**Algorithm 3** UPDATE_PSEUDO_REWARD
___
**Input:** $taskStack$, Parent's pseudo reward $\tilde{R}_p$
1: $tempCR \leftarrow \tilde{R}_p$, $N_{a'} \leftarrow 0$, $cr' \leftarrow 0$
2: **while** $taskStack$ is not empty **do**
3: $\quad \langle a, s, N_a, cr \rangle \leftarrow taskStack.pop()$
4: $\quad tempCR \leftarrow \gamma^{N_a'} \cdot tempCR + cr'$
5: $\quad$ Update pseudo-reward posterior $\Phi$ for $\tilde{R}(a, s)$ using $(a, s, tempCR)$
6: $\quad$ Resample $\tilde{R}(a, s)$ from $\Phi$
7: $\quad N_{a'} \leftarrow N_a$, $cr' \leftarrow cr$
8: **end while**
___

is arbitrarily worse than the hierarchically optimal policy. For all these reasons, PRs are major "nuisance parameters" in the MAXQ framework.

What makes learning PRs tricky is that they are not only value functions, but also function as *parameters* of MAXQ. That is, setting different PRs essentially results in a new learning problem. For this reason, simply trying to learn PRs in a standard temporal difference (TD) way fails (as we show in our experiments). Fortunately, Bayesian RL allows us to address both these issues. First, we can treat value functions as probabilistic unknown parameters. Second, and more importantly, a key idea in Bayesian RL is the "lifting" of exploration to the space of task parameters. That is, instead of exploration through action selection, Bayesian RL can perform exploration by sampling task parameters. Thus treating a PR as an unknown Bayesian parameter also leads to *exploration over the value of this parameter*, until an optimal value is found. In this way, hierarchically optimal policies can be learned from scratch—a major advantage over the standard MAXQ setting.

To learn PRs, we again maintain a distribution over all such parameters, $\Phi$, initially a prior. For simplicity, we only focus on tasks with multiple exits, since otherwise, a PR has no effect on the policy (though the value function changes). When a composite task executes, we keep track of each child task's execution in a stack. When the parent itself exits, we obtain a new observation of the PRs of each child by computing the discounted cumulative reward received *after* it exited, added to the current estimate of the parent's PR (Algorithm 3). This observation is used to update the current posterior over the child's PR. Since this is a value function estimate, early in the learning process, the estimates are noisy. Following prior work [8], we use a window containing the most recent observations. When a new observation arrives, the oldest observation is removed, the new one is added and a new posterior estimate is computed. After updating the posterior, it is sampled to obtain a new PR estimate for the associated exit. This estimate is used where needed (in Algorithms 1 and 2) until the next posterior update. Combined with the model-based priors above, we hypothesize that this procedure, iterated till convergence, will produce a hierarchically optimal policy.

## 4 Empirical Evaluation

In this section, we evaluate our approach and test four hypotheses: First, does incorporating model-based priors help speed up the convergence of MAXQ to the optimal policy? Second, does the task hierarchy still matter if very good priors are available for primitive actions? Third, how does Bayesian MAXQ compare to standard (flat) Bayesian RL? Does Bayesian RL perform better (in terms of computational time) if a task hierarchy is available? Finally, can our approach effectively learn PRs and policies that are hierarchically optimal?

We first focus on evaluating the first three hypotheses using domains where a zero PR results in hierarchical optimality. To evaluate these hypotheses, we use two domains: the fickle version of Taxi-World [5] (625 states) and Resource-collection [13] (8265 states). [2] In Taxi-World, the agent controls a taxi in a grid-world that has to pick up a passenger from a source location and drop them off at their destination. The state variables consist of the location of the taxi and the source and destination of the passenger. The actions available to the agent consist of navigation actions and actions to pickup and putdown the passenger. The agent gets a reward of $+20$ upon completing the task, a constant $-1$ reward for every action and a $-10$ penalty for an erroneous action. Further, each

___
[2]Task hierarchies for all domains are available in the supplementary material.

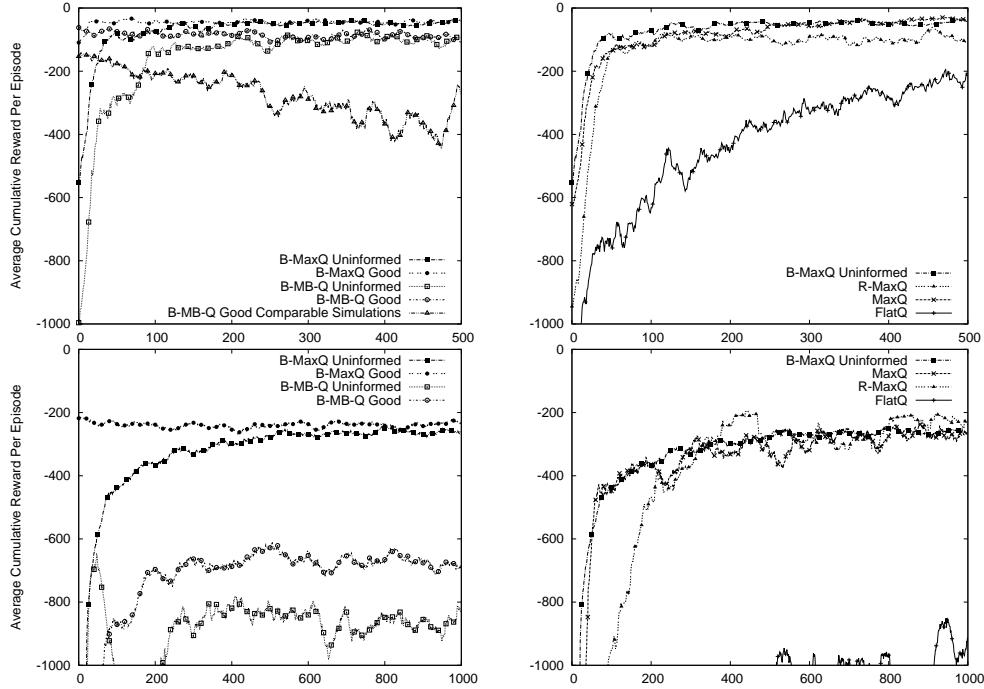

Figure 1: Performance on Taxi-World (top row) and Resource-collection (bottom). The x-axis shows episodes. The prefix "B-" denotes Bayesian, "Uninformed/Good" denotes the prior and "MB" denotes model-based. Left column: Bayesian methods, right: non-Bayesian methods, with Bayesian MAXQ for reference.

navigation action has a $15\%$ chance of moving in each direction orthogonal to the intended move. In the Resource-collection domain, the agent collects resources (gold and wood) from a grid world map. Here the state variables consist of the location of the agent, what the agent is carrying, whether a goldmine or forest is adjacent to its current location and whether a desired gold or wood quota has been met. The actions available to the agent are to move to a specific location, chop gold or harvest wood, and to deposit the item it is carrying (if any). For each navigation action, the agent has a $30\%$ chance of moving to a random location. In our experiments, the map contains two goldmines and two forests, each containing two units of gold and two units of wood, and the gold and wood quota is set to three each. The agent gets a +50 reward when it meets the gold/wood quota, a constant $-1$ reward for every action and an additional $-1$ for erroneous actions (such as trying to deposit when it is not carrying anything).

For the Bayesian methods, we use Dirichlet priors for the transition function parameters and Normal-Gamma priors for the reward function parameters. We use two priors: an uninformed prior, set to approximate a uniform distribution, and a "good" prior where a previously computed model posterior is used as the "prior." The prior distributions we use are conjugate to the likelihood, so we can compute the posterior distributions in closed form. In general, this is not necessary; more complex priors could be used as long as we can sample from the posterior distribution.

The methods we evaluate are: (i) Flat Q, the standard Q-learning algorithm, (ii) MAXQ-0, the standard, Q-learning algorithm for MAXQ with no PR, (iii) Bayesian model-based Q-learning with an uninformed prior and (iv) a "good" prior, (v) Bayesian MAXQ (our proposed approach) with an uninformed prior and (vi) a "good" prior, and (vii) RMAXQ [14]. In our implementation, the Bayesian model-based Q-learning uses the same code as the Bayesian MAXQ algorithm, with a "trivial" hierarchy consisting of the Root task with only the primitive actions as children. For the Bayesian methods, the update frequency $k$ was set to 50 for Taxi-World and 25 for Resource-collection. $Sim$ was set to 200 for Bayesian MAXQ for Taxi-World and 1000 for Bayesian model-based Q, and to 1000 for both for Resource collection. For RMAXQ, the threshold sample size $m$ was set to 5 following prior work [14]. The value iteration was terminated either after 300 loops or when the successive difference between iterations was less than 0.001. The theoretical version of RMAXQ requires updating and re-solving the model every step. In practice for the larger problems, this is too

time-consuming, so we re-solve the models every 10 steps. This is similar to the update frequency $k$ for Bayesian MAXQ. The results are shown in Figure 1 (episodes on x-axis).

From these results, comparing the Bayesian versions of MAXQ to standard MAXQ, we observe that for Taxi-World, the Bayesian version converges faster to the optimal policy even with the uninformed prior, while for Resource-collection, the convergence rates are similar. When a good prior is available, convergence is very fast (almost immediate) in both domains. Thus, the availability of model priors can help speed up convergence in many cases for HRL. We further observe that RMAXQ converges more slowly than MAXQ or Bayesian MAXQ, though it is much better than Flat Q. This is different from prior work [14]. This may be because our domains are more stochastic than the Taxi-world on which prior results [14] were obtained. We conjecture that, as the environment becomes more stochastic, errors in primitive model estimates may propagate into subtask models and hurt the performance of this algorithm. In their analysis [14], the authors noted that the error in the transition function for a composite task is a function of the total number of terminal states in the subtask. The error is also compounded as we move up the task hierarchy. This could be countered by increasing $m$, the sample size used to estimate model parameters. This would improve the accuracy of the primitive model, but would further hurt the convergence rate of the algorithm.

Next, we compare the Bayesian MAXQ approach to "flat" Bayesian model-based Q learning. We note that in Taxi-World, with uninformed priors, though the "flat" method initially does worse, it soon catches up to standard MAXQ and then to Bayesian MAXQ. This is probably because in this domain, the primitive models are relatively easy to acquire, and the task hierarchy provides no additional leverage. For Resource-collection, however, even with a good prior, "flat" Bayesian model-based Q does not converge. The difference is that in this case, the task hierarchy encodes extra information that cannot be deduced just from the models. In particular, the task hierarchy tells the agent that good policies consist of gold/wood collection moves followed by deposit moves. Since the reward structure in this domain is very sparse, it is difficult to deduce this even if very good models are available. Taken together, these results show that task hierarchies and model priors can be complementary: in general, Bayesian MAXQ outperforms both flat Bayesian RL and MAXQ (in speed of convergence, since here MAXQ can learn the hierarchically optimal policy).

Table 1: Time for 500 episodes, Taxi-World.

| Method | Time (s) |
|---|---|
| Bayesian MaxQ, Uninformed Prior | 205 |
| Bayesian Model-based Q, Uninformed Prior | 4684 |
| Bayesian MaxQ, Good Prior | 96 |
| Bayesian Model-based Q, Good Prior | 3089 |
| Bayesian Model-based Q, Good Prior & Comparable Simulations | 4006 |
| RMAXQ | 229 |
| MAXQ | 2.06 |
| Flat Q | 1.77 |

Next, we compare the time taken by the different approaches in our experiments in Taxi-World (Table 1). As expected, the Bayesian RL approaches are significantly slower than the non-Bayesian approaches. Further, among non-Bayesian approaches, the hierarchical approaches (MAXQ and RMAXQ) are slower than the non-hierarchical flat Q. Out of the Bayesian methods, however, the Bayesian MAXQ approaches are significantly faster than the flat Bayesian model-based approaches. This is because for the flat case, during the simulation in RECOMPUTE_VALUE, a much larger task needs to be solved, while the Bayesian MAXQ approach is able to take into account the structure of the hierarchy to only simulate subtasks as needed, which ends up being much more efficient. However, we note that we allowed the flat Bayesian model-based approach 1000 episodes of simulation as opposed to 200 for Bayesian MAXQ. Clearly this increases the time taken for the flat cases. But at the same time, this is necessary: the "Comparable Simulations" row (and curve in Figure 1 top left) shows that, if the simulations are reduced to 250 episodes for this approach, the resulting values are no longer reliable and the performance of the Bayesian flat approach drops sharply. Notice that while Flat Q runs faster than MAXQ (because of the additional "bookkeeping" overhead due to the task hierarchy), Bayesian MAXQ runs much faster than Bayesian model-based Q. Thus, taking advantage of the hierarchical task decomposition helps reduce the computational cost of Bayesian RL.

Finally we evaluate how well our approach estimates PRs. Here we use two domains: a Modified-Taxi-World and a Hallway domain [5, 21] (4320 states). In Modified-Taxi-World, we allow dropoffs at any one of the four locations and do not provide a reward for task termination. Thus the Navigate subtask needs a PR (corresponding to the correct dropoff location) to learn a good policy. The Hallway domain consists of a maze with a large scale structure of hallways and intersections. The agent has stochastic movement actions. For these experiments, we use uninformed priors on the environment model. The PR Gaussian-Gamma priors are set to prefer each exit from

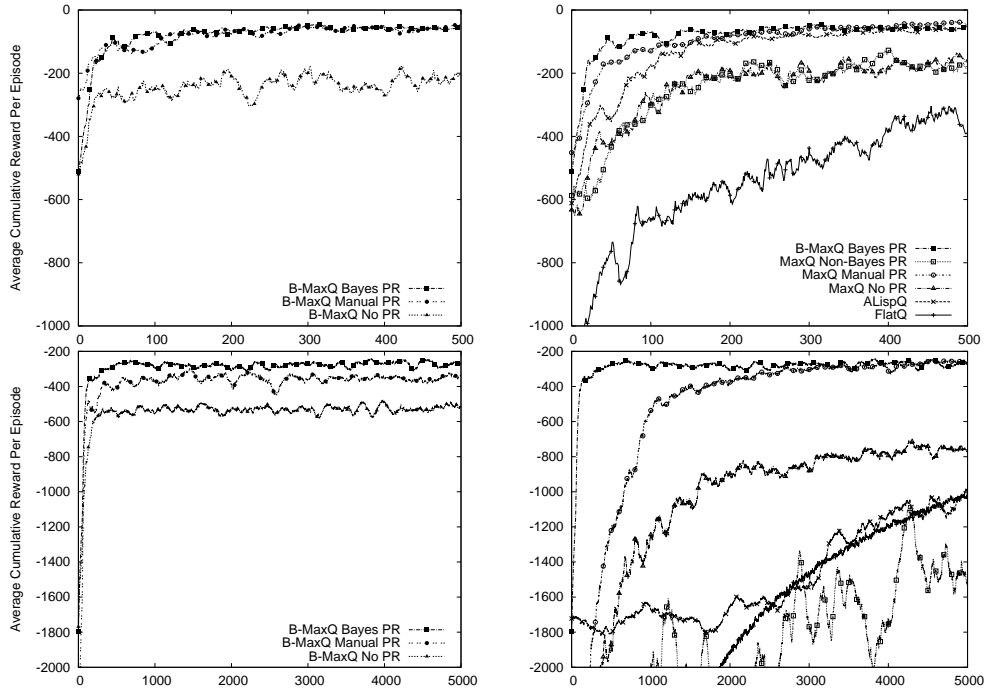

Figure 2: Performance on Modified-Taxi-World (top row) and Hallway (bottom). "B-": Bayesian, "PR": Pseudo Reward. Left: Bayesian methods, right: non-Bayesian methods, with Bayesian MAXQ as reference. The x-axis is episodes. The bottom right figure has the same legend as the top right.

a subtask equally. The baselines we use are: (i) Bayesian MAXQ and MAXQ with fixed zero PR, (ii) Bayesian MAXQ and MAXQ with fixed manually set PR, (iii) flat Q, (iv) ALISPQ [6] and (v) MAXQ with a non-Bayesian PR update. This last method tracks PR just as our approach; however, instead of a Bayesian update, it updates the PR using a temporal difference update, treating it as a simple value function. The results are shown in Figure 2 (episodes on x-axis).

From these results, we first observe that the methods with zero PR always do worse than those with "proper" PR, indicating that in these cases the recursively optimal policy is not the hierarchically optimal policy. When a PR is manually set, in both domain, MAXQ converges to better policies. We observe that in each case, the Bayesian MAXQ approach is able to learn a policy that is as good, starting with no pseudo rewards; further, its convergence rates are often better. Further, we observe that the simple TD update strategy (MAXQ Non-Bayes PR in Figure 2) fails in both cases—in Modified-Taxi-World, it is able to learn a policy that is approximately as good as a recursively optimal policy, but in the Hallway domain, it fails to converge completely, indicating that this strategy cannot generally learn PRs. Finally, we observe that the tripartite Q-decomposition of ALISPQ is also able to correctly learn hierarchically optimal policies, however, it converges slowly compared to Bayesian MAXQ or MAXQ with manual PRs. This is especially visible in the Hallway domain, where there are not many opportunities for state abstraction. We believe this is likely because it is estimating entire $Q$-functions rather than just the PRs. In a sense, it is doing more work than is needed to capture the hierarchically optimal policy, because an exact Q-function may not be needed to capture the preference for the best exit, rather, a value that assigns it a sufficiently high reward compared to the other exits would suffice. Taken together, these results indicate that incorporating Bayesian priors into MAXQ can successfully learn PRs from scratch and produce hierarchically optimal policies.

## 5 Conclusion

In this paper, we have proposed an approach to incorporating probabilistic priors on environment models and task pseudo-rewards into HRL by extending the MAXQ framework. Our experiments indicate that several synergies exist between HRL and Bayesian RL, and combining them is fruitful. In future work, we plan to investigate approximate model and value representations, as well as multi-task RL to learn the priors.

## Footnotes

[1]While we believe this description is accurate, unfortunately, due to language issues and some missing technical and experimental details in the cited article, we have been unable to replicate this work.

# References

[1] R.S. Sutton and A. G. Barto. *Reinforcement Learning: An Introduction*. MIT Press, 1998.

[2] Leslie Pack Kaelbling, Michael L. Littman, and Andrew W. Moore. Reinforcement learning: A survey. *Journal of Artificial Intelligence Research*, 4:237–285, 1996.

[3] Andrew G. Barto and Sridhar Mahadevan. Recent advances in hierarchical reinforcement learning. *Discrete Event Dynamic Systems*, 13(4):341–379, 2003.

[4] Martin Stolle and Doina Precup. *Learning Options in reinforcement Learning*, volume 2371/2002 of *Lecture Notes in Computer Science*, pages 212–223. Springer, 2002.

[5] Thomas G. Dietterich. Hierarchical reinforcement learning with the maxq value function decomposition. *Journal of Artificial Intelligence Research*, 13:227–303, 2000.

[6] D. Andre and S. Russell. State Abstraction for Programmable Reinforcement Learning Agents. In *Proceedings of the Eighteenth National Conference on Artificial Intelligence (AAAI)*, 2002.

[7] R. Dearden, N. Friedman, and D. Andre. Model based bayesian exploration. In *Proceedings of Fifteenth Conference on Uncertainty in Artificial Intelligence*. Morgan Kaufmann, 1999.

[8] R. Dearden, N. Friedman, and S. Russell. Bayesian Q-learning. In *Proceedings of the Fifteenth National Conference on Artificial Intelligence*, 1998.

[9] Y. Engel, S. Mannor, and R. Meir. Bayes meets Bellman:the Gaussian process approach to temporal difference learning. In *Proceedings of the Twentieth Internationl Conference on Machine Learning*, 2003.

[10] Mohammad Ghavamzadeh and Yaakov Engel. Bayesian policy gradient algorithms. In *Advances in Neural Information Processing Systems 19*. MIT Press, 2007.

[11] Alessandro Lazaric and Mohammad Ghavamzadeh. Bayesian multi-task reinforcement learning. In *Proceedings of the 27th International Conference on Machine Learning*, 2010.

[12] Aaron Wilson, Alan Fern, Soumya Ray, and Prasad Tadepalli. Multi-task reinforcement learning: a hierarchical bayesian approach. In *Proceedings of the 24th international conference on Machine learning*, pages 1015–1022, New York, NY, USA, 2007. ACM.

[13] N. Mehta, S. Ray, P. Tadepalli, and T. Dietterich. Automatic discovery and transfer of MAXQ hierarchies. In Andrew McCallum and Sam Roweis, editors, *Proceedings of the 25th International Conference on Machine Learning*, pages 648–655. Omnipress, 2008.

[14] Nicholas K. Jong and Peter Stone. Hierarchical model-based reinforcement learning: R-MAX + MAXQ. In *Proceedings of the 25th International Conference on Machine Learning*, 2008.

[15] Ronen I. Brafman, Moshe Tennenholtz, and Pack Kaelbling. R-MAX - a general polynomial time algorithm for near-optimal reinforcement learning. *Journal of Machine Learning Research*, 2001.

[16] B. Marthi, S. Russell, and D. Andre. A compact, hierarchically optimal q-function decomposition. In *22nd Conference on Uncertainty in Artificial Intelligence*, 2006.

[17] M. Ghavamzadeh and Y. Engel. Bayesian actor-critic algorithms. In Zoubin Ghahramani, editor, *Proceedings of the 24th Annual International Conference on Machine Learning*, pages 297–304. Omnipress, 2007.

[18] W. R. Thompson. On the likelihood that one unknown probability exceeds another in view of the evidence of two samples. *Biometrika*, 25:285–294, 1933.

[19] M. J. A. Strens. A Bayesian framework for reinforcement learning. In *Proceeding of the 17th International Conference on Machine Learning*, 2000.

[20] Zhaohui Dai, Xin Chen, Weihua Cao, and Min Wu. Model-based learning with bayesian and maxq value function decomposition for hierarchical task. In *Proceedings of the 8th World Congress on Intelligent Control and Automation*, 2010.

[21] Ronald Edward Parr. *Hierarchical Control and Learning for Markov Decision Processes*. PhD thesis, 1998.

